# Assessing Approximations for Gaussian Process Classification

**Malte Kuss** and **Carl Edward Rasmussen**
Max Planck Institute for Biological Cybernetics
Spemannstraße 38, 72076 Tübingen, Germany
{kuss,carl}@tuebingen.mpg.de

## Abstract

Gaussian processes are attractive models for probabilistic classification but unfortunately exact inference is analytically intractable. We compare Laplace's method and Expectation Propagation (EP) focusing on marginal likelihood estimates and predictive performance. We explain theoretically and corroborate empirically that EP is superior to Laplace. We also compare to a sophisticated MCMC scheme and show that EP is surprisingly accurate.

In recent years models based on Gaussian process (GP) priors have attracted much attention in the machine learning community. Whereas inference in the GP regression model with Gaussian noise can be done analytically, probabilistic classification using GPs is analytically intractable. Several approaches to approximate Bayesian inference have been suggested, including Laplace's approximation, Expectation Propagation (EP), variational approximations and Markov chain Monte Carlo (MCMC) sampling, some of these in conjunction with generalisation bounds, online learning schemes and sparse approximations.

Despite the abundance of recent work on probabilistic GP classifiers, most experimental studies provide only anecdotal evidence, and no clear picture has yet emerged, as to when and why which algorithm should be preferred. Thus, from a practitioners point of view probabilistic GP classification *remains a jungle*. In this paper, we set out to understand and compare two of the most wide-spread approximations: Laplace's method and Expectation Propagation (EP). We also compare to a sophisticated, but computationally demanding MCMC scheme to examine how close the approximations are to *ground truth*.

We examine two aspects of the approximation schemes: Firstly the accuracy of approximations to the marginal likelihood which is of central importance for model selection and model comparison. In any practical application of GPs in classification (usually multiple) parameters of the covariance function (hyperparameters) have to be handled. Bayesian model selection provides a consistent framework for setting such parameters. Therefore, it is essential to evaluate the accuracy of the marginal likelihood approximations as a function of the hyperparameters, in order to assess the practical usefulness of the approach

Secondly, we need to assess the quality of the approximate probabilistic predictions. In the past, the probabilistic nature of the GP predictions have not received much attention, the focus being mostly on classification error *rates*. This unfortunate state of affairs is caused primarily by typical benchmarking problems being considered outside of a realistic context. The ability of a classifier to produce class probabilities or confidences, have obvious

relevance in most areas of application, eg. medical diagnosis. We evaluate the predictive distributions of the approximate methods, and compare to the MCMC gold standard.

## 1 The Gaussian Process Model for Binary Classification

Let $y \in \{-1, 1\}$ denote the class label of an input $\mathbf{x}$. Gaussian process classification (GPC) is discriminative in modelling $p(y|\mathbf{x})$ for given $\mathbf{x}$ by a Bernoulli distribution. The probability of success $p(y=1|\mathbf{x})$ is related to an unconstrained latent function $f(\mathbf{x})$ which is mapped to the unit interval by a sigmoid transformation, eg. the *logit* or the *probit*. For reasons of analytic convenience we exclusively use the probit model $p(y=1|\mathbf{x}) = \Phi(f(\mathbf{x}))$, where $\Phi$ denotes the cumulative density function of the standard Normal distribution.

In the GPC model Bayesian inference is performed about the latent function $f$ in the light of observed data $\mathcal{D} = \{(y_i, \mathbf{x}_i)|i = 1, \ldots, m\}$. Let $f_i = f(\mathbf{x}_i)$ and $\mathbf{f} = [f_1, \ldots, f_m]^\top$ be shorthand for the values of the latent function and $\mathbf{y} = [y_1, \ldots, y_m]^\top$ and $\mathbf{X} = [\mathbf{x}_1, \ldots, \mathbf{x}_m]^\top$ collect the class labels and inputs respectively. Given the latent function the class labels are independent Bernoulli variables, so the joint likelihood factories:

$$p(\mathbf{y}|\mathbf{f}) = \prod_{i=1}^{m} p(y_i|f_i) = \prod_{i=1}^{m} \Phi(y_i f_i),$$

and depends on $f$ only through its value at the observed inputs. We use a zero-mean Gaussian process prior over the latent function $f$ with a covariance function $k(\mathbf{x}, \mathbf{x}'|\boldsymbol{\theta})$, which may depend on *hyperparameters* $\boldsymbol{\theta}$ [1]. The functional form and parameters of the covariance function encodes assumptions about the latent function, and adaptation of these is part of the inference. The posterior distribution over latent function values $\mathbf{f}$ at the observed $\mathbf{X}$ for given hyperparameters $\boldsymbol{\theta}$ becomes:

$$p(\mathbf{f}|\mathcal{D}, \boldsymbol{\theta}) = \frac{\mathcal{N}(\mathbf{f}|\mathbf{0}, \mathbf{K})}{p(\mathcal{D}|\boldsymbol{\theta})} \prod_{i=1}^{m} \Phi(y_i f_i), \quad \text{where} \quad p(\mathcal{D}|\boldsymbol{\theta}) = \int p(\mathbf{y}|\mathbf{f}) p(\mathbf{f}|\mathbf{X}, \boldsymbol{\theta}) d\mathbf{f},$$

denotes the marginal likelihood. Unfortunately neither the marginal likelihood, nor the posterior itself, or predictions can be computed analytically, so approximations are needed.

## 2 Approximate Bayesian Inference

For the GPC model approximations are either based on a Gaussian approximation to the posterior $p(\mathbf{f}|\mathcal{D}, \boldsymbol{\theta}) \approx q(\mathbf{f}|\mathcal{D}, \boldsymbol{\theta}) = \mathcal{N}(\mathbf{f}|\mathbf{m}, \mathbf{A})$ or involve Markov chain Monte Carlo (MCMC) sampling [2]. We compare Laplace's method and Expectation Propagation (EP) which are two alternative approaches to finding parameters $\mathbf{m}$ and $\mathbf{A}$ of the Gaussian $q(\mathbf{f}|\mathcal{D}, \boldsymbol{\theta})$. Both methods also allow approximate evaluation of the marginal likelihood, which is useful for ML-II hyperparameter optimisation.

Laplace's approximation (LA) is found by making a second order Taylor approximation of the (un-normalised) log posterior [3]. The mean $\mathbf{m}$ is placed at the mode (MAP) and the covariance $\mathbf{A}$ equals the negative inverse Hessian of the log posterior density at $\mathbf{m}$.

The EP approximation [4] also gives a Gaussian approximation to the posterior. The parameters $\mathbf{m}$ and $\mathbf{A}$ are found in an iterative scheme by matching the approximate marginal moments of $p(f_i|\mathcal{D}, \boldsymbol{\theta})$ by the marginals of the approximation $\mathcal{N}(f_i|\mathbf{m}_i, \mathbf{A}_{ii})$. Although we cannot *prove* the convergence of EP, we conjecture that it always converges for GPC with probit likelihood, and have never encountered an exception.

A key insight is that a Gaussian approximation to the GPC posterior is equivalent to a GP approximation to the posterior distribution over latent functions. For a test input $\mathbf{x}_*$ the

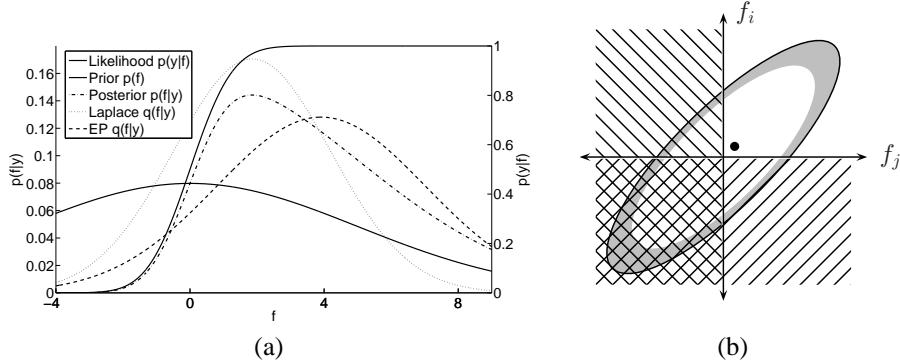

(a)                                                                    (b)

Figure 1: Panel (a) provides a one-dimensional illustration of the approximations. The prior $\mathcal{N}(f|0, 5^2)$ combined with the probit likelihood $(y = 1)$ results in a skewed posterior. The likelihood uses the right axis, all other curves use the left axis. Laplace's approximation peaks at the posterior mode, but places far too much mass over negative values of $f$ and too little at large positive values. The EP approximation matches the first two posterior moments, which results in a larger mean and a more accurate placement of probability mass compared to Laplace's approximation. In Panel (b) we caricature a high dimensional zero-mean Gaussian prior as an ellipse. The gray shadow indicates that for a high dimensional Gaussian most of the mass lies in a thin shell. For large latent signals (large entries in $\mathbf{K}$), the likelihood essentially cuts off regions which are incompatible with the training labels (hatched area), leaving the upper right orthant as the posterior. The dot represents the mode of the posterior, which remains close to the origin.

approximate predictive latent and class probabilities are:

$$q(f_*|\mathcal{D}, \boldsymbol{\theta}, \mathbf{x}_*) = \mathcal{N}(\mu_*, \sigma_*^2), \quad \text{and} \quad q(y_*=1|\mathcal{D}, \mathbf{x}_*) = \Phi(\mu_*/\sqrt{1 + \sigma_*^2}),$$

where $\mu_* = \mathbf{k}_*^\top \mathbf{K}^{-1} \mathbf{m}$ and $\sigma_*^2 = k(\mathbf{x}_*, \mathbf{x}_*) - \mathbf{k}_*^\top(\mathbf{K}^{-1} - \mathbf{K}^{-1}\mathbf{A}\mathbf{K}^{-1})\mathbf{k}_*$, where the vector $\mathbf{k}_* = [k(\mathbf{x}_1, \mathbf{x}_*), \ldots, k(\mathbf{x}_m, \mathbf{x}_*)]^\top$ collects covariances between $\mathbf{x}_*$ and training inputs $\mathbf{X}$.

MCMC sampling has the advantage that it becomes exact in the limit of long runs and so provides a *gold standard* by which to measure the two analytic methods described above. Although MCMC methods can in principle be used to do inference over $\mathbf{f}$ and $\boldsymbol{\theta}$ *jointly* [5], we compare to methods using ML-II optimisation over $\boldsymbol{\theta}$, thus we use MCMC to integrate over $\mathbf{f}$ only. Good marginal likelihood estimates are notoriously difficult to obtain; in our experiments we use Annealed Importance Sampling (AIS) [6], combining several Thermodynamic Integration runs into a single (unbiased) estimate of the marginal likelihood.

Both analytic approximations have a computational complexity which is cubic $\mathcal{O}(m^3)$ as common among non-sparse GP models due to inversions $m \times m$ matrices. In our implementations LA and EP need similar running times, on the order of a few minutes for several hundred data-points. Making AIS work efficiently requires some fine-tuning and a single estimate of $p(\mathcal{D}|\boldsymbol{\theta})$ can take several hours for data sets of a few hundred examples, but this could conceivably be improved upon.

## 3 Structural Properties of the Posterior and its Approximations

Structural properties of the posterior can best be understood by examining its construction. The prior is a correlated $m$-dimensional Gaussian $\mathcal{N}(\mathbf{f}|\mathbf{0}, \mathbf{K})$ centred at the origin. Each likelihood term $p(y_i|f_i)$ *softly* truncates the half-space from the prior that is incompatible with the observed label, see Figure 1. The resulting posterior is *unimodal* and *skewed*, similar to a multivariate Gaussian truncated to the orthant containing $\mathbf{y}$. The mode of

the posterior remains close to the origin, while the mass is placed in accordance with the observed class labels. Additionally, high dimensional Gaussian distributions exhibit the property that most probability mass is contained in a thin ellipsoidal shell – depending on the covariance structure – away from the mean [7, ch. 29.2]. Intuitively this occurs since in high dimensions the volume grows extremely rapidly with the radius. As an effect the mode becomes less representative (typical) for the prior distribution as the dimension increases. For the GPC posterior this property persists: the mode of the posterior distribution stays relatively close to the origin, still being unrepresentative for the posterior distribution, while the mean moves to the mass of the posterior making mean and mode differ significantly.

We cannot generally assume the posterior to be close to Gaussian, as in the often studied limit of low-dimensional parametric models with large amounts of data. Therefore in GPC we must be aware of making a Gaussian approximation to a non-Gaussian posterior. From the properties of the posterior it can be expected that Laplace's method places $\mathbf{m}$ in the right orthant but too close to the origin, such that the approximation will overlap with regions having practically zero posterior mass. As an effect the amplitude of the approximate latent posterior GP will be underestimated systematically, leading to overly cautious predictive distributions. The EP approximation does not rely on a local expansion, but assumes that the marginal distributions can be well approximated by Gaussians. This assumption will be examined empirically below.

## 4 Experiments

In this section we compare and inspect approximations for GPC using various benchmark data sets. The primary focus is not to optimise the absolute performance of GPC models but to compare the relative accuracy of approximations and to validate the arguments given in the previous section. In all experiments we use a covariance function of the form:

$$k(\mathbf{x}, \mathbf{x}'|\boldsymbol{\theta}) = \sigma^2 \exp\left(-\tfrac{1}{2}\left\|\mathbf{x} - \mathbf{x}'\right\|^2 / \ell^2\right), \tag{1}$$

such that $\boldsymbol{\theta} = [\sigma, \ell]$. We refer to $\sigma^2$ as the signal variance and to $\ell$ as the characteristic length-scale. Note that for many classification tasks it may be reasonable to use an individual length scale parameter for every input dimension (ARD) or a different kind of covariance function. Nevertheless, for the sake of presentability we use the above covariance function and we believe the conclusions about the accuracy of approximations to be independent of this choice, since it relies on arguments which are independent of the form of the covariance function.

As measure of the accuracy of predictive probabilities we use the average information in bits of the predictions about the test targets in excess of that of random guessing. Let $p^* = p(y_* = 1|\mathcal{D}, \boldsymbol{\theta}, \mathbf{x}_*)$ be the model's prediction, then we average:

$$I(p_i^*, y_i) = \tfrac{y_i+1}{2}\log_2(p_i^*) + \tfrac{1-y_i}{2}\log_2(1 - p_i^*) + H \tag{2}$$

over all test cases, where $H$ is the entropy of the training labels. The error rate $E$ is equal to the percentage of erroneous class assignments if prediction is understood as a decision problem with symmetric costs.

For the first set of experiments presented here the well-known USPS digits and the Ionosphere data set were used. A binary sub-problem from the USPS digits is defined by only considering 3's vs. 5's (which is probably the hardest of the binary sub-problems) and dividing the data into 767 cases for training and 773 for testing. The Ionosphere data is split into 200 training and 151 test cases. We do an exhaustive investigation on a fine regular grid of values for the log hyperparameters. For each $\boldsymbol{\theta}$ on the grid we compute the approximated log marginal likelihood by LA, EP and AIS. Additionally we compute the respective predictive performance (2) on the test set. Results are shown in Figure 2.

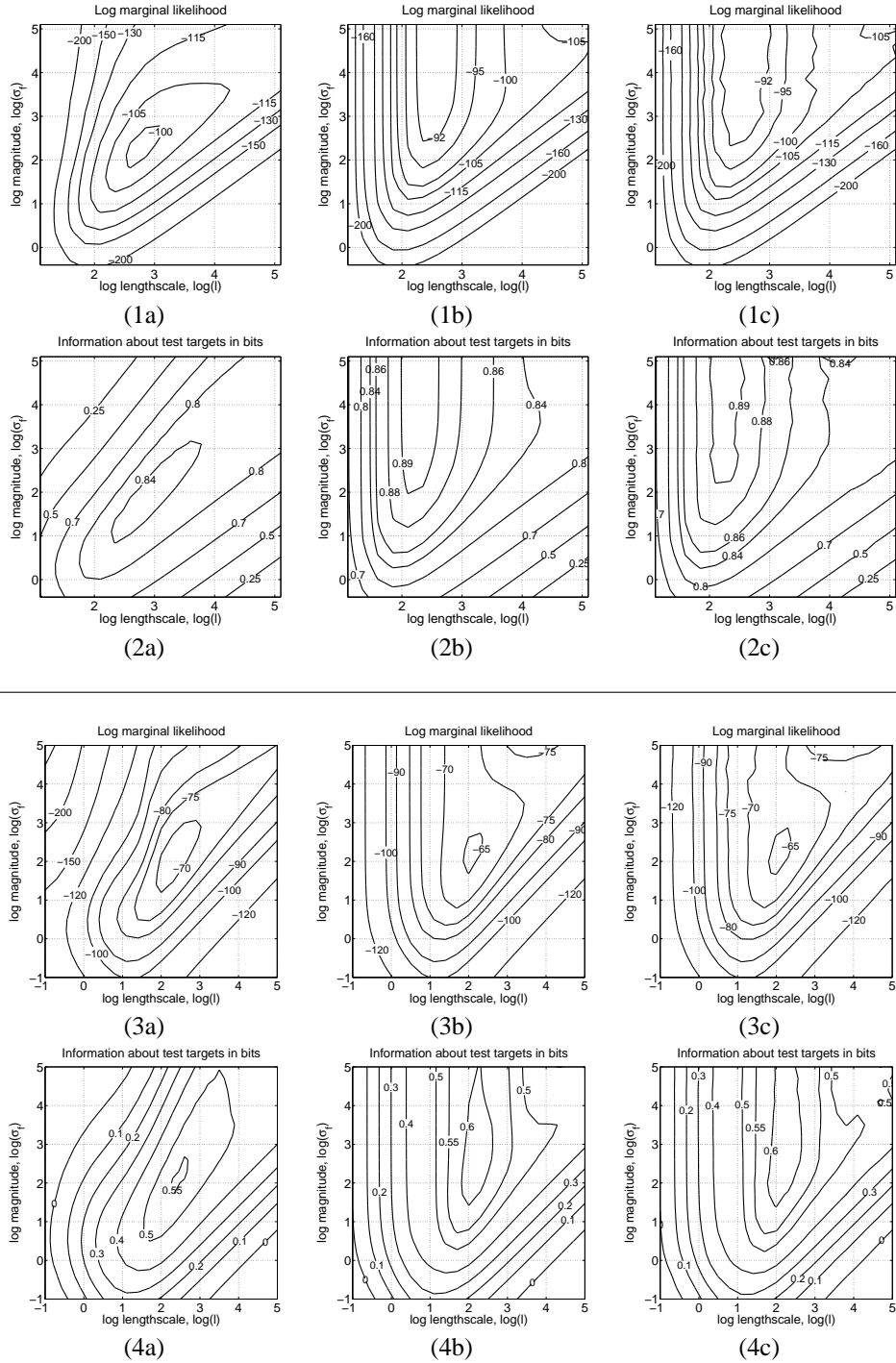

Figure 2: Comparison of marginal likelihood approximations and predictive performances of different approximation techniques for USPS 3s vs. 5s (upper half) and the Ionosphere data (lower half). The columns correspond to LA (a), EP (b), and MCMC (c). The rows show estimates of the log marginal likelihood (rows 1 & 3) and the corresponding predictive performance (2) on the test set (rows 2 & 4) respectively.

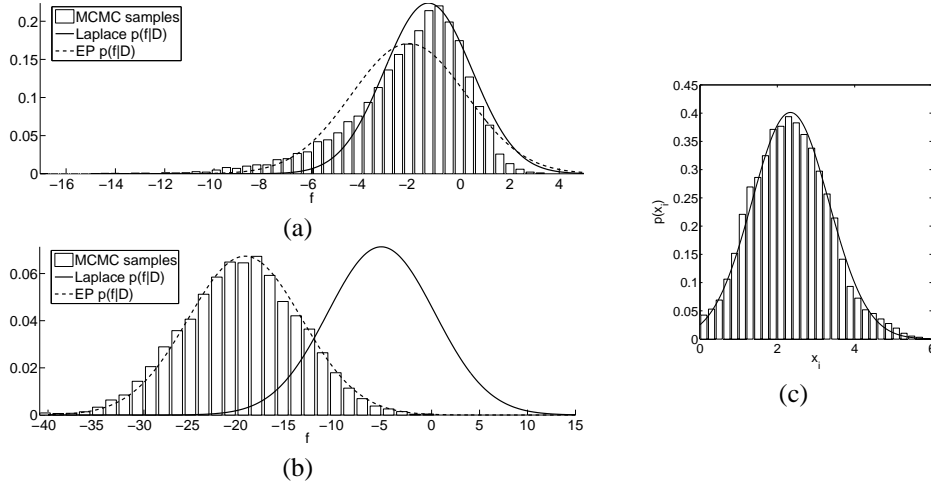

Figure 3: Panel (a) and (b) show two marginal distributions $p(f_i|\mathcal{D}, \boldsymbol{\theta})$ from a GPC posterior and its approximations. The true posterior is approximated by a normalised histogram of 9000 samples of $f_i$ obtained by MCMC sampling. Panel (c) shows a histogram of samples of a marginal distribution of a truncated high-dimensional Gaussian. The line describes a Gaussian with mean and variance estimated from the samples.

For all three approximation techniques we see an agreement between marginal likelihood estimates and test performance, which justifies the use of ML-II parameter estimation. But the shape of the contours and the values differ between the methods. The contours for Laplace's method appear to be *slanted* compared to EP. The marginal likelihood estimates of EP and AIS agree surprisingly well[1], given that the marginal likelihood comes as a $767$ respectively $200$ dimensional integral. The EP predictions contain as much information about the test cases as the MCMC predictions and significantly more than for LA. Note that for small signal variances (roughly $\ln(\sigma^2) < 1$) LA and EP give very similar results. A possible explanation is that for small signal variances the likelihood does not *truncate* the prior but only *down-weights* the tail that disagrees with the observation. As an effect the posterior will be less skewed and both approximations will lead to similar results.

For the USPS 3's vs. 5's we now inspect the marginal distributions $p(f_i|\mathcal{D}, \boldsymbol{\theta})$ of single latent function values under the posterior approximations for a given value of $\boldsymbol{\theta}$. We have chosen the values $\ln(\sigma) = 3.35$ and $\ln(\ell) = 2.85$ which are between the ML-II estimates of EP and LA. Hybrid MCMC was used to generate 9000 samples from the posterior $p(\mathbf{f}|\mathcal{D}, \boldsymbol{\theta})$. For LA and EP the approximate marginals are $q(f_i|\mathcal{D}, \boldsymbol{\theta}) = \mathcal{N}(f_i|\mathbf{m}_i, \mathbf{A}_{ii})$ where $\mathbf{m}$ and $\mathbf{A}$ are found by the respective approximation techniques.

In general we observe that the marginal distributions of MCMC samples agree very well with the respective marginal distributions of the EP approximation. For Laplace's approximation we find the mean to be underestimated and the marginal distributions to overlap with zero far more than the EP approximations. Figure (3a) displays the marginal distribution and its approximations for which the MCMC samples show maximal skewness. Figure (3b) shows a typical example where the EP approximation agrees very well with the MCMC samples. We show this particular example because under the EP approximation $p(y_i = 1|\mathcal{D}, \boldsymbol{\theta}) < 0.1\%$ but LA gives a wrong $p(y_i = 1|\mathcal{D}, \boldsymbol{\theta}) \approx 18\%$.

In the experiment we saw that the marginal distributions of the posterior often agree very

well with a Gaussian approximation. This seems to contradict the description given in the previous section were we argued that the posterior is skewed by construction. In order to inspect the marginals of a truncated high-dimensional multivariate Gaussian distribution we made an additional synthetic experiment. We constructed a 767 dimensional Gaussian $\mathcal{N}(\mathbf{x}|\mathbf{0}, \mathbf{C})$ with a covariance matrix having one eigenvalue of 100 with eigenvector $\mathbf{1}$, and all other eigenvalues are 1. We then truncate this distribution such that all $\mathbf{x}_i \geq 0$. Note that the mode of the truncated Gaussian is still at zero, whereas the mean moves towards the remaining mass. Figure (3c) shows a normalised histogram of samples from a marginal distribution of one $\mathbf{x}_i$. The samples agree very well with a Gaussian approximation. In the previous section we described the somewhat surprising property, that for a truncated high-dimensional Gaussian, resembling the posterior, the mode (used by LA) may not be particularly representative of the distribution. Although the marginal is also truncated, it is still exceptionally well modelled by a Gaussian – however, the Laplace approximation centred on the origin would be completely inappropriate.

In a second set of experiments we compare the predictive performance of LA and EP for GPC on several well known benchmark problems. Each data set is randomly split into 10 folds of which one at a time is left out as a test set to measure the predictive performance of a model trained (or selected) on the remaining nine folds. All performance measures are averages over the 10 folds. For GPC we implement model selection by ML-II hyperparameter estimation, reporting results given the $\boldsymbol{\theta}$ that maximised the respective approximate marginal likelihoods $p(\mathcal{D}|\boldsymbol{\theta})$.

In order to get a better picture of the absolute performance we also compare to results obtained by C-SVM classification. The kernel we used is equivalent to the covariance function (1) without the signal variance parameter. For each fold the parameters $C$ and $\ell$ are found in an inner loop of 5-fold cross-validation, in which the parameter grids are refined until the performance stabilises. Predictive probabilities for test cases are obtained by mapping the unthresholded output of the SVM to $[0, 1]$ using a sigmoid function [8].

Results are summarised in Table 1. Comparing Laplace's method to EP the latter shows to be more accurate both in terms of error rate and information. While the error rates are relatively similar the predictive distribution obtained by EP shows to be more informative about the test targets. Note that for GPC the error rate only depends of the sign of the mean $\mu_*$ of the approximated posterior over latent functions and not the entire posterior predictive distribution. As to be expected, the length of the mean vector $\|\mathbf{m}\|$ shows much larger values for the EP approximations. Comparing EP and SVMs the results are mixed. For the Crabs data set all methods show the same error rate but the information content of the predictive distributions differs dramatically. For some test cases the SVM predicts the wrong class with large certainty.

## 5   Summary & Conclusions

Our experiments reveal serious differences between Laplace's method and EP when used in GPC models. From the structural properties of the posterior we described why LA systematically underestimates the mean $\mathbf{m}$. The resulting posterior GP over latent functions will have too small amplitude, although the sign of the mean function will be mostly correct. As an effect LA gives over-conservative predictive probabilities, and diminished information about the test labels. This effect has been show empirically on several real world examples. Large resulting discrepancies in the actual posterior probabilities were found, even at the training locations, which renders the predictive class probabilities produced under this approximation grossly inaccurate. Note, the difference becomes less dramatic if we only consider the classification error rates obtained by thresholding $p^*$ at $1/2$. For this particular task, we've seen the the sign of the latent function tends to be correct (at least at the training locations).

| Data Set | m | n | Laplace E% | Laplace I | Laplace $\|\mathbf{m}\|$ | EP E% | EP I | EP $\|\mathbf{m}\|$ | SVM E% | SVM I |
|---|---|---|---|---|---|---|---|---|---|---|
| Ionosphere | 351 | 34 | 8.84 | 0.591 | 49.96 | 7.99 | 0.661 | 124.94 | 5.69 | 0.681 |
| Wisconsin | 683 | 9 | 3.21 | 0.804 | 62.62 | 3.21 | 0.805 | 84.95 | 3.21 | 0.795 |
| Pima Indians | 768 | 8 | 22.77 | 0.252 | 29.05 | 22.63 | 0.253 | 47.49 | 23.01 | 0.232 |
| Crabs | 200 | 7 | 2.0 | 0.682 | 112.34 | 2.0 | 0.908 | 2552.97 | 2.0 | 0.047 |
| Sonar | 208 | 60 | 15.36 | 0.439 | 26.86 | 13.85 | 0.537 | 15678.55 | 11.14 | 0.567 |
| USPS 3 vs 5 | 1540 | 256 | 2.27 | 0.849 | 163.05 | 2.21 | 0.902 | 22011.70 | 2.01 | 0.918 |

Table 1: Results for benchmark data sets. The first three columns give the name of the data set, number of observations $m$ and dimension of inputs $n$. For Laplace's method and EP the table reports the average error rate E%, the average information I (2) and the average length $\|\mathbf{m}\|$ of the mean vector of the Gaussian approximation. For SVMs the error rate and the average information about the test targets are reported. Note that for the Crabs data set we use the sex (not the colour) of the crabs as class label.

The EP approximation has shown to give results very close to MCMC both in terms of predictive distributions and marginal likelihood estimates. We have shown and explained why the marginal distributions of the posterior can be well approximated by Gaussians.

Further, the marginal likelihood values obtained by LA and EP differ systematically which will lead to different results of ML-II hyperparameter estimation. The discrepancies are similar for different tasks. Using AIS we were able to show the accuracy of marginal likelihood estimates, which to the best of our knowledge has never been done before.

In summary, we found that EP is the method of choice for approximate inference in binary GPC models, when the computational cost of MCMC is prohibitive. In contrast, the Laplace approximation is so inaccurate that we advise against its use, especially when predictive probabilities are to be taken seriously. Further experiments and a detailed description of the approximation schemes can be found in [2].

**Acknowledgements**   Both authors acknowledge support by the German Research Foundation (DFG) through grant RA 1030/1. This work was supported in part by the IST Programme of the European Community, under the PASCAL Network of Excellence, IST-2002-506778. This publication only reflects the authors' views.

## Footnotes

[1]Note that the agreement between the two seems to be limited by the accuracy of the MCMC runs, as judged by the regularity of the contour lines; the tolerance is less than one unit on a (natural) log scale.

# References

[1] C. K. I. Williams and C. E. Rasmussen. Gaussian processes for regression. In David S. Touretzky, Michael C. Mozer, and Michael E. Hasselmo, editors, *NIPS 8*, pages 514–520. MIT Press, 1996.

[2] M. Kuss and C. E. Rasmussen. Assessing approximate inference for binary Gaussian process classification. *Journal of Machine Learning Research*, 6:1679–1704, 2005.

[3] C. K. I. Williams and D. Barber. Bayesian classification with Gaussian processes. *IEEE Transactions on Pattern Analysis and Machine Intelligence*, 20(12):1342–1351, 1998.

[4] T. P. Minka. *A Family of Algorithms for Approximate Bayesian Inference*. PhD thesis, Department of Electrical Engineering and Computer Science, MIT, 2001.

[5] R. M. Neal. Regression and classification using Gaussian process priors. In J. M. Bernardo, J. O. Berger, A. P. Dawid, and A. F. M. Smith, editors, *Bayesian Statistics 6*, pages 475–501. Oxford University Press, 1998.

[6] R. M. Neal. Annealed importance sampling. *Statistics and Computing*, 11:125–139, 2001.

[7] D. J. C. MacKay. *Information Theory, Inference and Learning Algorithms*. CUP, 2003.

[8] J. C. Platt. Probabilities for SV machines. In *Advances in Large Margin Classifiers*, pages 61–73. The MIT Press, 2000.